# Energy Disaggregation via Discriminative Sparse Coding

**J. Zico Kolter**
Computer Science and
Artificial Intelligence Laboratory
Massachusetts Institute of Technology
Cambridge, MA 02139
kolter@csail.mit.edu

**Siddarth Batra, Andrew Y. Ng**
Computer Science Department
Stanford University
Stanford, CA 94305
{sidbatra,ang}@cs.stanford.edu

## Abstract

Energy disaggregation is the task of taking a whole-home energy signal and sep-
arating it into its component appliances. Studies have shown that having device-
level energy information can cause users to conserve significant amounts of en-
ergy, but current electricity meters only report whole-home data. Thus, developing
algorithmic methods for disaggregation presents a key technical challenge in the
effort to maximize energy conservation. In this paper, we examine a large scale
energy disaggregation task, and apply a novel extension of sparse coding to this
problem. In particular, we develop a method, based upon structured prediction,
for discriminatively training sparse coding algorithms specifically to maximize
disaggregation performance. We show that this significantly improves the perfor-
mance of sparse coding algorithms on the energy task and illustrate how these
disaggregation results can provide useful information about energy usage.

## 1 Introduction

Energy issues present one of the largest challenges facing our society. The world currently consumes
an average of 16 terawatts of power, 86% of which comes from fossil fuels [28]; without any effort
to curb energy consumption or use different sources of energy, most climate models predict that the
earth's temperature will increase by at least 5 degrees Fahrenheit in the next 90 years [1], a change
that could cause ecological disasters on a global scale. While there are of course numerous facets to
the energy problem, there is a growing consensus that many energy and sustainability problems are
fundamentally informatics problems, areas where machine learning can play a significant role.

This paper looks specifically at the task of energy disaggregation, an informatics task relating to
energy efficiency. Energy disaggregation, also called non-intrusive load monitoring [11], involves
taking an aggregated energy signal, for example the total power consumption of a house as read by
an electricity meter, and separating it into the different electrical appliances being used. Numerous
studies have shown that receiving information about ones energy usage can automatically induce
energy-conserving behaviors [6, 19], and these studies also clearly indicate that receiving appliance-
specific information leads to much larger gains than whole-home data alone ([19] estimates that
appliance-level data could reduce consumption by an average of 12% in the residential sector). In
the United States, electricity constitutes 38% of all energy used, and residential and commercial
buildings together use 75% of this electricity [28]; thus, this 12% figure accounts for a sizable
amount of energy that could potentially be saved. However, the widely-available sensors that provide
electricity consumption information, namely the so-called "Smart Meters" that are already becoming
ubiquitous, collect energy information only at the whole-home level and at a very low resolution
(typically every hour or 15 minutes). Thus, energy disaggregation methods that can take this whole-
home data and use it to predict individual appliance usage present an algorithmic challenge where
advances can have a significant impact on large-scale energy efficiency issues.

Energy disaggregation methods do have a long history in the engineering community, including some which have applied machine learning techniques — early algorithms [11, 26] typically looked for "edges" in power signal to indicate whether a known device was turned on or off; later work focused on computing harmonics of steady-state power or current draw to determine more complex device signatures [16, 14, 25, 2]; recently, researchers have analyzed the transient noise of an electrical circuit that occurs when a device changes state [15, 21]. However, these and all other studies we are aware of were either conducted in artificial laboratory environments, contained a relatively small number of devices, trained and tested on the same set of devices in a house, and/or used custom hardware for very high frequency electrical monitoring with an algorithmic focus on "event detection" (detecting when different appliances were turned on and off). In contrast, in this paper we focus on disaggregating electricity using low-resolution, hourly data of the type that is readily available via smart meters (but where most single-device "events" are not apparent); we specifically look at the generalization ability of our algorithms for devices and homes unseen at training time; and we consider a data set that is substantially larger than those previously considered, with 590 homes, 10,165 unique devices, and energy usage spanning a time period of over two years.

The algorithmic approach we present in this paper builds upon sparse coding methods and recent work in single-channel source separation [24, 23, 22]. Specifically, we use a sparse coding algorithm to learn a model of each device's power consumption over a typical week, then combine these learned models to predict the power consumption of different devices in previously unseen homes, using their aggregate signal alone. While energy disaggregation can naturally be formulated as such a single-channel source separation problem, we know of no previous application of these methods to the energy disaggregation task. Indeed, the most common application of such algorithm is audio signal separation, which typically has very high temporal resolution; thus, the low-resolution energy disaggregation task we consider here poses a new set of challenges for such methods, and existing approaches alone perform quite poorly.

As a second major contribution of the paper, we develop a novel approach for discriminatively training sparse coding dictionaries for disaggregation tasks, and show that this significantly improves performance on our energy domain. Specifically, we formulate the task of maximizing disaggregation performance as a structured prediction problem, which leads to a simple and effective algorithm for discriminatively training such sparse representation for disaggregation tasks. The algorithm is similar in spirit to a number of recent approaches to discriminative training of sparse representations [12, 17, 18]. However, these past works were interested in discriminatively training sparse coding representation specifically for *classification* tasks, whereas we focus here on discriminatively training the representation for *disaggregation* tasks, which naturally leads to substantially different algorithmic approaches.

## 2  Discriminative Disaggregation via Sparse Coding

We begin by reviewing sparse coding methods and their application to disaggregation tasks. For concreteness we use the terminology of our energy disaggregation domain throughout this description, but the algorithms can apply equally to other domains. Formally, assume we are given $k$ different classes, which in our setting corresponds to device categories such as televisions, refrigerators, heaters, etc. For every $i = 1, \ldots, k$, we have a matrix $\mathbf{X}_i \in \mathbb{R}^{T \times m}$ where each *column* of $\mathbf{X}_i$ contains a week of energy usage (measured every hour) for a particular house and for this particular type of device. Thus, for example, the $j$th column of $\mathbf{X}_1$, which we denote $\mathbf{x}_1^{(j)}$, may contain weekly energy consumption for a refrigerator (for a single week in a single house) and $\mathbf{x}_2^{(j)}$ could contain weekly energy consumption of a heater (for this same week in the same house). We denote the *aggregate* power consumption over all device types as $\bar{\mathbf{X}} \equiv \sum_{i=1}^{k} \mathbf{X}_i$ so that the $j$th column of $\bar{\mathbf{X}}$, $\bar{\mathbf{x}}^{(j)}$, contains a week of aggregated energy consumption for all devices in a given house. At training time, we assume we have access to the individual device energy readings $\mathbf{X}_1, \ldots, \mathbf{X}_k$ (obtained for example from plug-level monitors in a small number of instrumented homes). At test time, however, we assume that we have access only to the aggregate signal of a new set of data points $\bar{\mathbf{X}}'$ (as would be reported by smart meter), and the goal is to separate this signal into its components, $\mathbf{X}_1', \ldots, \mathbf{X}_k'$.

The sparse coding approach to source separation (e.g., [24, 23]), which forms for the basis for our disaggregation approach, is to train separate models for each individual class $\mathbf{X}_i$, then use these models to separate an aggregate signal. Formally, sparse coding models the $i$th data matrix using the approximation $\mathbf{X}_i \approx \mathbf{B}_i \mathbf{A}_i$ where the columns of $\mathbf{B}_i \in \mathbb{R}^{T \times n}$ contain a set of $n$ basis functions, also called the *dictionary*, and the columns of $\mathbf{A}_i \in \mathbb{R}^{n \times m}$ contain the *activations* of these basis functions

[20]. Sparse coding additionally imposes the the constraint that the activations $\mathbf{A}_i$ be sparse, i.e., that they contain mostly zero entries, which allows us to learn *overcomplete* representations of the data (more basis functions than the dimensionality of the data). A common approach for achieving this sparsity is to add an $\ell_1$ regularization penalty to the activations.

Since energy usage is an inherently non-negative quantity, we impose the further constraint that the activations and bases be non-negative, an extension known as non-negative sparse coding [13, 7]. Specifically, in this paper we will consider the non-negative sparse coding objective

$$\min_{\mathbf{A}_i \geq 0, \mathbf{B}_i \geq 0} \frac{1}{2} \|\mathbf{X}_i - \mathbf{B}_i \mathbf{A}_i\|_F^2 + \lambda \sum_{p,q} (\mathbf{A}_i)_{pq} \quad \text{subject to} \quad \|\mathbf{b}_i^{(j)}\|_2 \leq 1, \ j = 1, \dots, n \quad (1)$$

where $\mathbf{X}_i$, $\mathbf{A}_i$, and $\mathbf{B}_i$ are defined as above, $\lambda \in \mathbb{R}_+$ is a regularization parameter, $\|\mathbf{Y}\|_F \equiv (\sum_{p,q} Y_{pq})^{1/2}$ is the Frobenius norm, and $\|\mathbf{y}\|_2 \equiv (\sum_p y_p^2)^{1/2}$ is the $\ell_2$ norm. This optimization problem is not jointly convex in $\mathbf{A}_i$ and $\mathbf{B}_i$, but it is convex in each optimization variable when holding the other fixed, so a common strategy for optimizing (1) is to alternate between minimizing the objective over $\mathbf{A}_i$ and $\mathbf{B}_i$.

After using the above procedure to find representations $\mathbf{A}_i$ and $\mathbf{B}_i$ for each of the classes $i = 1, \dots, k$, we can disaggregate a new aggregate signal $\bar{\mathbf{X}} \in \mathbb{R}^{T \times m'}$ (without providing the algorithm its individual components), using the following procedure (used by, e.g., [23], amongst others). We concatenate the bases to form single joint set of basis functions and solve the optimization problem

$$\hat{\mathbf{A}}_{1:k} = \arg \min_{\mathbf{A}_{1:k} \geq 0} \left\| \bar{\mathbf{X}} - [\mathbf{B}_1 \cdots \mathbf{B}_k] \begin{bmatrix} \mathbf{A}_1 \\ \vdots \\ \mathbf{A}_k \end{bmatrix} \right\|_F^2 + \lambda \sum_{i,p,q} (\mathbf{A}_i)_{pq}$$

$$\equiv \arg \min_{\mathbf{A}_{1:k} \geq 0} F(\bar{\mathbf{X}}, \mathbf{B}_{1:k}, \mathbf{A}_{1:k}) \quad (2)$$

where for ease of notation we use $\mathbf{A}_{1:k}$ as shorthand for $\mathbf{A}_1, \dots, \mathbf{A}_k$, and we abbreviate the optimization objective as $F(\bar{\mathbf{X}}, \mathbf{B}_{1:k}, \mathbf{A}_{1:k})$. We then predict the $i$th component of the signal to be

$$\hat{\mathbf{X}}_i = \mathbf{B}_i \hat{\mathbf{A}}_i. \quad (3)$$

The intuition behind this approach is that if $\mathbf{B}_i$ is trained to reconstruct the $i$th class with small activations, then it should be better at reconstructing the $i$th portion of the aggregate signal (i.e., require smaller activations) than all other bases $\mathbf{B}_j$ for $j \neq i$. We can evaluate the quality of the resulting disaggregation by what we refer to as the *disaggregation error*,

$$E(\mathbf{X}_{1:k}, \mathbf{B}_{1:k}) \equiv \sum_{i=1}^k \frac{1}{2} \|\mathbf{X}_i - \mathbf{B}_i \hat{\mathbf{A}}_i\|_F^2 \ \text{ subject to } \ \hat{\mathbf{A}}_{1:k} = \arg \min_{\mathbf{A}_{1:k} \geq 0} F \left( \sum_{i=1}^k \mathbf{X}_i, \mathbf{B}_{1:k}, \mathbf{A}_{1:k} \right),$$

$$(4)$$

which quantifies how accurately we reconstruct each individual class when using the activations obtained only via the aggregated signal.

## 2.1 Structured Prediction for Discriminative Disaggregation Sparse Coding

An issue with using sparse coding alone for disaggregation tasks is that the bases are not trained to minimize the disaggregation error. Instead, the method relies on the hope that learning basis functions for each class individually will produce bases that are distinct enough to also produce small disaggregation error. Furthermore, it is very difficult to optimize the disaggregation error directly over $\mathbf{B}_{1:k}$, due to the non-differentiability (and discontinuity) of the argmin operator with a non-negativity constraint. One could imagine an alternating procedure where we iteratively optimize over $\mathbf{B}_{1:k}$, ignoring the the dependence of $\hat{\mathbf{A}}_{1:k}$ on $\mathbf{B}_{1:k}$, then re-solve for the activations $\hat{\mathbf{A}}_{1:k}$; but ignoring how $\hat{\mathbf{A}}_{1:k}$ depends on $\mathbf{B}_{1:k}$ loses much of the problem's structure and this approach performs very poorly in practice. Alternatively, other methods (though in a different context from disaggregation) have been proposed that use a differentiable objective function and implicit differentiation to explicitly model the derivative of the activations with respect to the basis functions [4]; however, this formulation loses some of the benefits of the standard sparse coding formulation, and computing these derivatives is a computationally expensive procedure.

Instead, we propose in this paper a method for optimizing disaggregation performance based upon structured prediction methods [27]. To describe our approach, we first define the *regularized disaggregation error*, which is simply the disaggregation error plus a regularization penalty on $\hat{\mathbf{A}}_{1:k}$,

$$E_{reg}(\mathbf{X}_{1:k}, \mathbf{B}_{1:k}) \equiv E(\mathbf{X}_{1:k}, \mathbf{B}_{1:k}) + \lambda \sum_{i,p,q} (\hat{\mathbf{A}}_i)_{pq} \tag{5}$$

where $\hat{\mathbf{A}}$ is defined as in (2). This criterion provides a better optimization objective for our algorithm, as we wish to obtain a *sparse* set of coefficients that can achieve low disaggregation error. Clearly, the best possible value of $\hat{\mathbf{A}}_i$ for this objective function is given by

$$\mathbf{A}_i^\star = \arg \min_{\mathbf{A}_i \geq 0} \frac{1}{2} \|\mathbf{X}_i - \mathbf{B}_i \mathbf{A}_i\|_F^2 + \lambda \sum_{p,q} (\mathbf{A}_i)_{pq}, \tag{6}$$

which is precisely the activations obtained after an iteration of sparse coding on the data matrix $\mathbf{X}_i$. Motivated by this fact, the first intuition of our algorithm is that in order to minimize disaggregation error, we can *discriminatively* optimize the bases $\mathbf{B}_{1:k}$ that such performing the optimization (2) produces activations that are as close to $\mathbf{A}_{1:k}^\star$ as possible. Of course, changing the bases $\mathbf{B}_{1:k}$ to optimize this criterion would also change the resulting optimal coefficients $\mathbf{A}_{1:k}^\star$. Thus, the second intuition of our method is that the bases used in the optimization (2) *need not be the same* as the bases used to reconstruct the signals. We define an augmented regularized disaggregation error objective

$$\tilde{E}_{reg}(\mathbf{X}_{1:k}, \mathbf{B}_{1:k}, \tilde{\mathbf{B}}_{1:k}) \equiv \sum_{i=1}^k \left( \frac{1}{2} \|\mathbf{X}_i - \mathbf{B}_i \hat{\mathbf{A}}_i\|_F^2 + \lambda \sum_{p,q} (\hat{\mathbf{A}}_i)_{pq} \right)$$
$$\text{subject to} \quad \hat{\mathbf{A}}_{1:k} = \arg \min_{\mathbf{A}_{1:k} \geq 0} F \left( \sum_{i=1}^k \mathbf{X}_i, \tilde{\mathbf{B}}_{1:k}, \mathbf{A}_{1:k} \right), \tag{7}$$

where the $\mathbf{B}_{1:k}$ bases (referred to as the *reconstruction bases*) are the same as those learned from sparse coding while the $\tilde{\mathbf{B}}_{1:k}$ bases (refereed to as the *disaggregation bases*) are discriminatively optimized in order to move $\hat{\mathbf{A}}_{1:k}$ closer to $\mathbf{A}_{1:k}^\star$, without changing these targets.

Discriminatively training the disaggregation bases $\tilde{\mathbf{B}}_{1:k}$ is naturally framed as a structured prediction task: the input is $\bar{\mathbf{X}}$, the multi-variate desired output is $\mathbf{A}_{1:k}^\star$, the model parameters are $\tilde{\mathbf{B}}_{1:k}$, and the discriminant function is $F(\bar{\mathbf{X}}, \tilde{\mathbf{B}}_{1:k}, \mathbf{A}_{1:k})$.[1] In other words, we seek bases $\tilde{\mathbf{B}}_{1:k}$ such that (ideally)

$$\mathbf{A}_{1:k}^\star = \arg \min_{\mathbf{A}_{1:k} \geq 0} F(\bar{\mathbf{X}}, \tilde{\mathbf{B}}_{1:k}, \mathbf{A}_{1:k}). \tag{8}$$

While there are many potential methods for optimizing such a prediction task, we use a simple method based on the structured perceptron algorithm [5]. Given some value of the parameters $\tilde{\mathbf{B}}_{1:k}$, we first compute $\hat{\mathbf{A}}$ using (2). We then perform the perceptron update with a step size $\alpha$,

$$\tilde{\mathbf{B}}_{1:k} \leftarrow \tilde{\mathbf{B}}_{1:k} - \alpha \left( \nabla_{\tilde{\mathbf{B}}_{1:k}} F(\bar{\mathbf{X}}, \tilde{\mathbf{B}}_{1:k}, \mathbf{A}_{1:k}^\star) - \nabla_{\tilde{\mathbf{B}}_{1:k}} F(\bar{\mathbf{X}}, \tilde{\mathbf{B}}_{1:k}, \hat{\mathbf{A}}_{1:k}) \right) \tag{9}$$

or more explicitly, defining $\tilde{\mathbf{B}} = \left[ \tilde{\mathbf{B}}_1 \cdots \tilde{\mathbf{B}}_k \right]$, $\mathbf{A}^\star = \left[ \mathbf{A}_1^{\star T} \cdots \mathbf{A}_1^{\star T} \right]^T$ (and similarly for $\hat{\mathbf{A}}$),

$$\tilde{\mathbf{B}} \leftarrow \tilde{\mathbf{B}} - \alpha \left( (\bar{\mathbf{X}} - \tilde{\mathbf{B}}\hat{\mathbf{A}})\hat{\mathbf{A}}^T - (\bar{\mathbf{X}} - \tilde{\mathbf{B}}\mathbf{A}^\star)\mathbf{A}^{\star T} \right). \tag{10}$$

To keep $\tilde{\mathbf{B}}_{1:k}$ in a similar form to $\mathbf{B}_{1:k}$, we keep only the positive part of $\tilde{\mathbf{B}}_{1:k}$ and we re-normalize each column to have unit norm. One item to note is that, unlike typical structured prediction where the discriminant is a *linear* function in the parameters (which guarantees convexity of the problem), here our discriminant is a *quadratic* function of the parameters, and so we no longer expect to necessarily reach a global optimum of the prediction problem; however, since sparse coding itself is a non-convex problem, this is not overly concerning for our setting. Our complete method for discriminative disaggregation sparse coding, which we call DDSC, is shown in Algorithm 1.

**Algorithm 1** Discriminative disaggregation sparse coding

---
**Input:** data points for each individual source $\mathbf{X}_i \in \mathbb{R}^{T \times m}$, $i = 1, \ldots, k$, regularization parameter $\lambda \in \mathbb{R}_+$, gradient step size $\alpha \in \mathbb{R}_+$.

**Sparse coding pre-training:**

    1. Initialize $\mathbf{B}_i$ and $\mathbf{A}_i$ with positive values and scale columns of $\mathbf{B}_i$ such that $\|\mathbf{b}_i^{(j)}\|_2 = 1$.

    2. For each $i = 1, \ldots, k$, iterate until convergence:

        (a) $\mathbf{A}_i \leftarrow \arg\min_{\mathbf{A} \geq 0} \|\mathbf{X}_i - \mathbf{B}_i\mathbf{A}\|_F^2 + \lambda \sum_{p,q} \mathbf{A}_{pq}$

        (b) $\mathbf{B}_i \leftarrow \arg\min_{\mathbf{B} \geq 0, \|\mathbf{b}^{(j)}\|_2 \leq 1} \|\mathbf{X}_i - \mathbf{B}\mathbf{A}_i\|_F^2$

**Discriminative disaggregation training:**

    3. Set $\mathbf{A}_{1:k}^{\star} \leftarrow \mathbf{A}_{1:k}$, $\tilde{\mathbf{B}}_{1:k} \leftarrow \mathbf{B}_{1:k}$.

    4. Iterate until convergence:

        (a) $\hat{\mathbf{A}}_{1:k} \leftarrow \arg\min_{\mathbf{A}_{1:k} \geq 0} F(\bar{\mathbf{X}}, \tilde{\mathbf{B}}_{1:k}, \mathbf{A}_{1:k})$

        (b) $\tilde{\mathbf{B}} \leftarrow \left[ \tilde{\mathbf{B}} - \alpha \left( (\bar{\mathbf{X}} - \tilde{\mathbf{B}}\hat{\mathbf{A}})\hat{\mathbf{A}}^T - (\bar{\mathbf{X}} - \tilde{\mathbf{B}}\mathbf{A}^{\star})(\mathbf{A}^{\star})^T \right) \right]_+$

        (c) For all $i, j$, $\mathbf{b}_i^{(j)} \leftarrow \mathbf{b}_i^{(j)}/\|\mathbf{b}_i^{(j)}\|_2$.

**Given aggregated test examples $\bar{\mathbf{X}}'$:**

    5. $\hat{\mathbf{A}}_{1:k}' \leftarrow \arg\min_{\mathbf{A}_{1:k} \geq 0} F(\bar{\mathbf{X}}', \tilde{\mathbf{B}}_{1:k}, \mathbf{A}_{1:k})$

    6. Predict $\hat{\mathbf{X}}_i' = \mathbf{B}_i\hat{\mathbf{A}}_i'$.

---

## 2.2 Extensions

Although, as we show shortly, the discriminative training procedure has made the largest difference in terms of improving disaggregation performance in our domain, a number of other modifications to the standard sparse coding formulation have also proven useful. Since these are typically trivial extensions or well-known algorithms, we mention them only briefly here.

**Total Energy Priors.** One deficiency of the sparse coding framework for energy disaggregation is that the optimization objective does not take into consideration the size of an energy signal for determining which class it belongs to, just its shape. Since total energy used is obviously a discriminating factor for different device types, we consider an extension that penalizes the $\ell_2$ deviation between a device and its mean total energy. Formally, we augment the objective $F$ with the penalty

$$F_{TEP}(\bar{\mathbf{X}}, \mathbf{B}_{1:k}, \mathbf{A}_{1:k}) = F(\bar{\mathbf{X}}, \mathbf{B}_{1:k}, \mathbf{A}_{1:k}) + \lambda_{TEP} \sum_{i=1}^{k} \|\mu_i \mathbf{1}^T - \mathbf{1}^T \mathbf{B}_i\mathbf{A}_i\|_2^2 \qquad (11)$$

where $\mathbf{1}$ denotes a vector of ones of the appropriate size, and $\mu_i = \frac{1}{m}\mathbf{1}^T\mathbf{X}_i$ denotes the average total energy of device class $i$.

**Group Lasso.** Since the data set we consider exhibits some amount of sparsity at the device level (i.e., several examples have zero energy consumed by certain device types, as there is either no such device in the home or it was not being monitored), we also would like to encourage a *grouping* effect to the activations. That is, we would like a certain coefficient being active for a particular class to encourage other coefficients to also be active in that class. To achieve this, we employ the group Lasso algorithm [29], which adds an $\ell_2$ norm penalty to the activations of each device

$$F_{GL}(\bar{\mathbf{X}}, \mathbf{B}_{1:k}, \mathbf{A}_{1:k}) = F(\bar{\mathbf{X}}, \mathbf{B}_{1:k}, \mathbf{A}_{1:k}) + \lambda_{GL} \sum_{i=1}^{k} \sum_{j=1}^{m} \|\mathbf{a}_i^{(j)}\|_2. \qquad (12)$$

**Shift Invariant Sparse Coding.** Shift invariant, or convolutional sparse coding is an extension to the standard sparse coding framework where each basis is convolved over the input data, with a separate activation for each shift position [3, 10]. Such a scheme may intuitively seem to be beneficial for the energy disaggregation task, where a given device might exhibit the same energy signature at different times. However, as we will show in the next section, this extension actually perform worse in our domain; this is likely due to the fact that, since we have ample training data

and a relatively low-dimensional domain (each energy signal has 168 dimensions, 24 hours per day times 7 days in the week), the standard sparse coding bases are able to cover all possible shift positions for typical device usage. However, pure shift invariant bases *cannot* capture information about *when* in the week or day each device is typically used, and such information has proven crucial for disaggregation performance.

## 2.3 Implementation

Space constraints preclude a full discussion of the implementation details of our algorithms, but for the most part we rely on standard methods for solving the optimization problems. In particular, most of the time spent by the algorithm involves solving sparse optimization problems to find the activation coefficients, namely steps 2a and 4a in Algorithm 1. We use a coordinate descent approach here, both for the standard and group Lasso version of the optimization problems, as these have been recently shown to be efficient algorithms for $\ell_1$-type optimization problems [8, 9], and have the added benefit that we can warm-start the optimization with the solution from previous iterations. To solve the optimization over $\mathbf{B}_i$ in step 2b, we use the multiplicative non-negative matrix factorization update from [7].

# 3  Experimental Results

## 3.1  The Plugwise Energy Data Set and Experimental Setup

We conducted this work using a data set provided by Plugwise, a European manufacturer of plug-level monitoring devices. The data set contains hourly energy readings from 10,165 different devices in 590 homes, collected over more than two years. Each device is labeled with one of 52 device types, which we further reduce to ten broad categories of electrical devices: lighting, TV, computer, other electronics, kitchen appliances, washing machine and dryer, refrigerator and freezer, dishwasher, heating/cooling, and a miscellaneous category. We look at time periods in blocks of one week, and try to predict the individual device consumption over this week given only the whole-home signal (since the data set does not currently contain true whole-home energy readings, we approximate the home's overall energy usage by aggregating the individual devices). Crucially, we focus on disaggregating data from homes that are absent from the training set (we assigned 70% of the homes to the training set, and 30% to the test set, resulting in 17,133 total training weeks and 6846 testing weeks); thus, we are attempting to generalize over the basic category of devices, not just over different uses of the same device in a single house. We fit the hyper-parameters of the algorithms (number of bases and regularization parameters) using grid search over each parameter independently on a cross validation set consisting of 20% of the training homes.

## 3.2  Qualitative Evaluation of the Disaggregation Algorithms

We first look qualitatively at the results obtained by the method. Figure 1 shows the true energy energy consumed by two different houses in the test set for two different weeks, along with the energy consumption predicted by our algorithms. The figure shows both the predicted energy of several devices over the whole week, as well as a pie chart that shows the relative energy consumption of different device types over the whole week (a more intuitive display of energy consumed over the week). In many cases, certain devices like the refrigerator, washer/dryer, and computer are predicted quite accurately, both in terms the total predicted percentage and in terms of the signals themselves. There are also cases where certain devices are not predicted well, such as underestimating the heating component in the example on the left, and a predicting spike in computer usage in the example on the right when it was in fact a dishwasher. Nonetheless, despite some poor predictions at the hourly device level, the breakdown of electric consumption is still quite informative, determining the approximate percentage of many devices types and demonstrating the promise of such feedback.

In addition to the disaggregation results themselves, sparse coding representations of the different device types are interesting in their own right, as they give a good intuition about how the different devices are typically used. Figure 2 shows a graphical representation of the learned basis functions. In each plot, the grayscale image on the right shows an intensity map of all bases functions learned for that device category, where each column in the image corresponds to a learned basis. The plot on the left shows examples of seven basis functions for the different device types. Notice, for example, that the bases learned for the washer/dryer devices are nearly all heavily peaked, while the refrigerator bases are much lower in maximum magnitude. Additionally, in the basis images devices like lighting demonstrate a clear "band" pattern, indicating that these devices are likely to

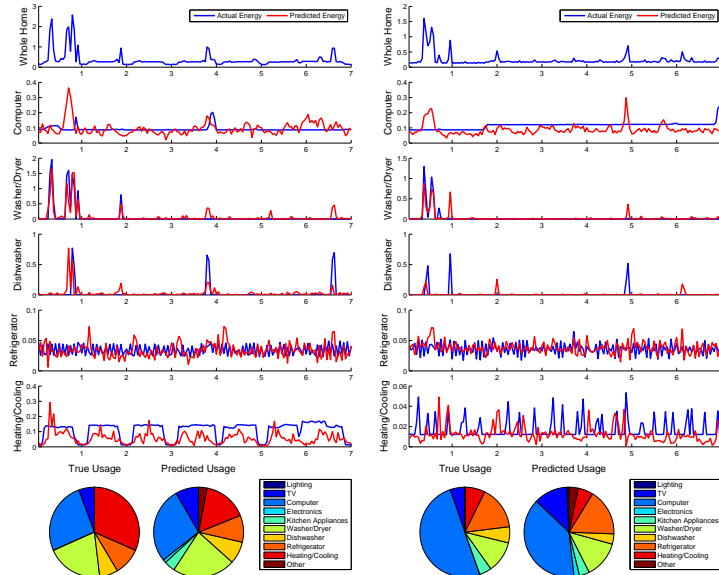

Figure 1: Example predicted energy profiles and total energy percentages (best viewed in color). Blue lines show the true energy usage, and red the predicted usage, both in units of kWh.

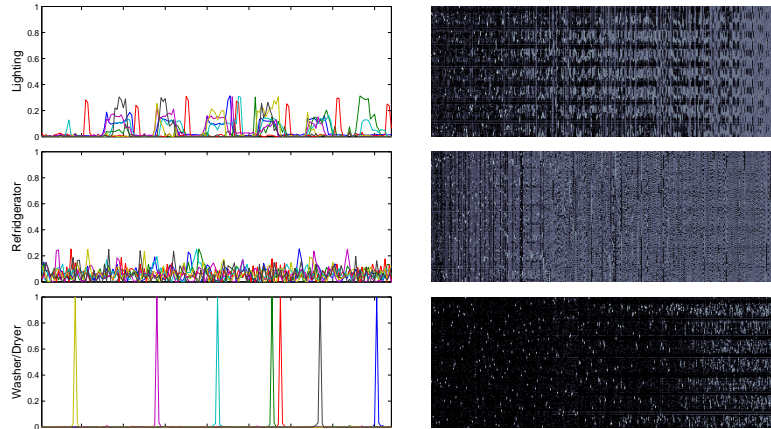

Figure 2: Example basis functions learned from three device categories (best viewed in color). The plot of the left shows seven example bases, while the image on the right shows all learned basis functions (one basis per column).

be on and off during certain times of the day (each basis covers a week of energy usage, so the seven bands represent the seven days). The plots also suggests why the standard implementation of shift invariance is not helpful here. There is sufficient training data such that, for devices like washers and dryers, we learn a separate basis for all possible shifts. In contrast, for devices like lighting, where the *time* of usage is an important factor, simple shift-invariant bases miss key information.

### 3.3 Quantitative Evaluation of the Disaggregation Methods

There are a number of components to the final algorithm we have proposed, and in this section we present quantitative results that evaluate the performance of each of these different components. While many of the algorithmic elements improve the disaggregation performance, the results in this section show that the discriminative training in particular is crucial for optimizing disaggregation performance. The most natural metric for evaluating disaggregation performance is the disaggregation error in (4). However, average disaggregation error is not a particularly intuitive metric, and so we also evaluate a total-week accuracy of the prediction system, defined formally as

$$\text{Accuracy} \equiv \frac{\sum_{i,q} \min\left\{\sum_p (\mathbf{X}_i)_{pq}, \sum_p (\mathbf{B}_i \hat{\mathbf{A}}_i)_{pq}\right\}}{\sum_{p,q} \bar{\mathbf{X}}_{p,q}}. \tag{13}$$

| Method | Training Set | | Test Accuracy | |
|---|---|---|---|---|
| | Disagg. Err. | Acc. | Disagg. Err. | Acc. |
| Predict Mean Energy | 20.98 | 45.78% | 21.72 | 47.41% |
| SISC | 20.84 | 41.87% | 24.08 | 41.79% |
| Sparse Coding | 10.54 | 56.96% | 18.69 | 48.00% |
| Sparse Coding + TEP | 11.27 | 55.52% | 16.86 | 50.62% |
| Sparse Coding + GL | 10.55 | 54.98% | 17.18 | 46.46% |
| Sparse Coding + TEP + GL | 9.24 | 58.03% | 14.05 | 52.52% |
| DDSC | 7.20 | 64.42% | 15.59 | 53.70% |
| DDSC + TEP | 8.99 | 59.61% | 15.61 | 53.23% |
| DDSC + GL | 7.59 | 63.09% | 14.58 | 52.20% |
| DDSC + TEP + GL | 7.92 | 61.64% | **13.20** | **55.05%** |

Table 1: Disaggregation results of algorithms (TEP = Total Energy Prior, GL = Group Lasso, SISC = Shift Invariant Sparse Coding, DDSC = Discriminative Disaggregation Sparse Coding).

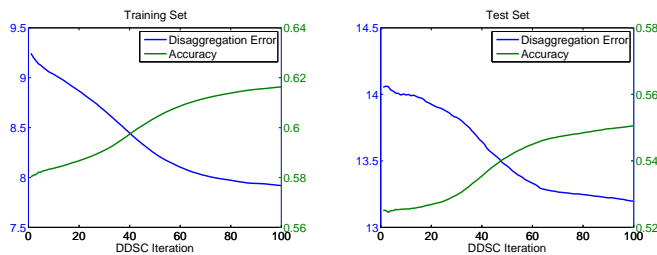

Figure 3: Evolution of training and testing errors for iterations of the discriminative DDSC updates.

Despite the complex definition, this quantity simply captures the average amount of energy predicted correctly over the week (i.e., the overlap between the true and predicted energy pie charts).

Table 1 shows the disaggregation performance obtained by many different prediction methods. The advantage of the discriminative training procedure is clear: all the methods employing discriminative training perform nearly as well or better than all the methods without discriminative training; furthermore, the system with all the extensions, discriminative training, a total energy prior, and the group Lasso, outperforms all competing methods on both metrics. To put these accuracies in context, we note that separate to the results presented here we trained an SVM, using a variety of hand-engineered features, to *classify* individual energy signals into their device category, and were able to achieve at most 59% classification accuracy. It therefore seems unlikely that we could disaggregate a signal to above this accuracy and so, informally speaking, we expect the achievable performance on this particular data set to range between 47% for the baseline of predicting mean energy (which in fact is a very reasonable method, as devices often follow their average usage patterns) and 59% for the individual classification accuracy. It is clear, then, that the discriminative training is crucial to improving the performance of the sparse coding disaggregation procedure within this range, and does provide a significant improvement over the baseline. Finally, as shown in Figure 3, both the training and testing error decrease reliably with iterations of DDSC, and we have found that this result holds for a wide range of parameter choices and step sizes (though, as with all gradient methods, some care be taken to choose a step size that is not prohibitively large).

## 4   Conclusion

Energy disaggregation is a domain where advances in machine learning can have a significant impact on energy use. In this paper we presented an application of sparse coding algorithms to this task, focusing on a large data set that contains the type of low-resolution data readily available from smart meters. We developed the discriminative disaggregation sparse coding (DDSC) algorithm, a novel discriminative training procedure, and show that this algorithm significantly improves the accuracy of sparse coding for the energy disaggregation task.

**Acknowledgments** This work was supported by ARPA-E (Advanced Research Projects Agency–Energy) under grant number DE-AR0000018. We are very grateful to Plugwise for providing us with their plug-level energy data set, and in particular we thank Willem Houck for his assistance with this data. We also thank Carrie Armel and Adrian Albert for helpful discussions.

## Footnotes

[1]The structured prediction task actually involves $m$ examples (where $m$ is the number of columns of $\bar{\mathbf{X}}$), and the goal is to output the desired activations $(\mathbf{a}_{1:k}^\star)^{(j)}$, for the $j$th example $\bar{\mathbf{x}}^{(j)}$. However, since the function $F$ decomposes across the columns of $\mathbf{X}$ and $\mathbf{A}$, the above notation is equivalent to the more explicit formulation.

# References

[1] D. Archer. *Global Warming: Understanding the Forecast*. Blackwell Publishing, 2008.

[2] M. Berges, E. Goldman, H. S. Matthews, and L Soibelman. Learning systems for electric comsumption of buildings. In *ASCI International Workshop on Computing in Civil Engineering*, 2009.

[3] T. Blumensath and M. Davies. On shift-invariant sparse coding. *Lecture Notes in Computer Science*, 3195(1):1205–1212, 2004.

[4] D. Bradley and J.A. Bagnell. Differentiable sparse coding. In *Advances in Neural Information Processing Systems*, 2008.

[5] M. Collins. Discriminative training methods for hidden markov models: Theory and experiements with perceptron algorithms. In *Proceedings of the Conference on Empirical Methods in Natural Language Processing*, 2002.

[6] S. Darby. The effectiveness of feedback on energy consumption. Technical report, Environmental Change Institute, University of Oxford, 2006.

[7] J. Eggert and E. Korner. Sparse coding and NMF. In *IEEE International Joint Conference on Neural Networks*, 2004.

[8] J. Friedman, T. Hastie, H Hoefling, and R. Tibshirani. Pathwise coordinate optimization. *The Annals of Applied Statistics*, 2(1):302–332, 2007.

[9] J. Friedman, T. Hastie, and R. Tibshirani. A note on the group lasso and a sparse group lasso. Technical report, Stanford University, 2010.

[10] R. Grosse, R. Raina, H. Kwong, and A. Y. Ng. Shift-invariant sparse coding for audio classification. In *Proceedings of the Conference on Uncertainty in Artificial Intelligence*, 2007.

[11] G. Hart. Nonintrusive appliance load monitoring. *Proceedings of the IEEE*, 80(12), 1992.

[12] S. Hasler, H. Wersin, and E Korner. Combining reconstruction and discrimination with class-specific sparse coding. *Neural Computation*, 19(7):1897–1918, 2007.

[13] P.O. Hoyer. Non-negative sparse coding. In *IEEE Workshop on Neural Networks for Signal Processing*, 2002.

[14] C. Laughman, K. Lee, R. Cox, S. Shaw, S. Leeb, L. Norford, and P. Armstrong. Power signature analysis. *IEEE Power & Energy Magazine*, 2003.

[15] C. Laughman, S. Leeb, and Lee. Advanced non-intrusive monitoring of electric loads. *IEEE Power and Energy*, 2003.

[16] W. Lee, G. Fung, H. Lam, F. Chan, and M. Lucente. Exploration on load signatures. *International Conference on Electrical Engineering (ICEE)*, 2004.

[17] J. Mairal, F. Bach, J. Ponce, G. Sapiro, and A. Zisserman. Supervised dictionary learning. In *Advances in Neural Information Processing Systems*, 2008.

[18] J. Mairal, M. Leordeanu, F. Bach, M. Hebert, and J. Ponce. Discriminative sparse image models for class-specific edge detection and image interpretation. In *European Conference on Computer Vision*, 2008.

[19] B. Neenan and J. Robinson. Residential electricity use feedback: A research synthesis and economic framework. Technical report, Electric Power Research Institute, 2009.

[20] B. A. Olshausen and D. J. Field. Emergence of simple-cell receptive field properties by learning a sparse code for natural images. *Nature*, 381:607–609, 1996.

[21] S. N. Patel, T. Robertson, J. A. Kientz, M. S. Reynolds, and G. D. Abowd. At the flick of a switch: Detecting and classifying unique electrical events on the residential power line. *9th international conference on Ubiquitous Computing (UbiComp 2007)*, 2007.

[22] S. T. Roweis. One microphone source separation. In *Advances in Neural Information Processing Systems*, 2000.

[23] M. N. Schmidt, J. Larsen, and F. Hsiao. Wind noise reduction using non-negative sparse coding. In *IEEE Workshop on Machine Learning for Signal Processing*, 2007.

[24] M N. Schmidt and R. K. Olsson. Single-channel speech separation using sparse non-negative matrix factorization. In *International Conference on Spoken Language Processing*, 2006.

[25] S. R. Shaw, C. B. Abler, R. F. Lepard, D. Luo, S. B. Leeb, and L. K. Norford. Instrumentation for high performance nonintrusive electrical load monitoring. *ASME*, 120(224), 1998.

[26] F. Sultanem. Using appliance signatures for monitoring residential loads at meter panel level. *IEEE Transaction on Power Delivery*, 6(4), 1991.

[27] B. Taskar, V. Chatalbashev, D. Koller, and C. Guestrin. Learning structured prediction models: A large margin approach. In *International Conference on Machine Learning*, 2005.

[28] Various. *Annual Energy Review 2009*. U.S. Energy Information Administration, 2009.

[29] M. Yuan and Y. Lin. Model selection and estimation in regression with grouped variables. *Journal of the Royal Statisical Society, Series B*, 68(1):49–67, 2007.

